# Extracting Tree-Structured Representations of Trained Networks

**Mark W. Craven and Jude W. Shavlik**
Computer Sciences Department
University of Wisconsin-Madison
1210 West Dayton St.
Madison, WI 53706
craven@cs.wisc.edu, shavlik@cs.wisc.edu

## Abstract

A significant limitation of neural networks is that the representations they learn are usually incomprehensible to humans. We present a novel algorithm, TREPAN, for extracting comprehensible, symbolic representations from trained neural networks. Our algorithm uses queries to induce a decision tree that approximates the concept represented by a given network. Our experiments demonstrate that TREPAN is able to produce decision trees that maintain a high level of fidelity to their respective networks while being comprehensible and accurate. Unlike previous work in this area, our algorithm is general in its applicability and scales well to large networks and problems with high-dimensional input spaces.

## 1 Introduction

For many learning tasks, it is important to produce classifiers that are not only highly accurate, but also easily understood by humans. Neural networks are limited in this respect, since they are usually difficult to interpret after training. In contrast to neural networks, the solutions formed by "symbolic" learning systems (e.g., Quinlan, 1993) are usually much more amenable to human comprehension. We present a novel algorithm, TREPAN, for extracting comprehensible, symbolic representations from trained neural networks. TREPAN queries a given network to induce a decision tree that describes the concept represented by the network. We evaluate our algorithm using several real-world problem domains, and present results that demonstrate that TREPAN is able to produce decision trees that are accurate and comprehensible, and maintain a high level of fidelity to the networks from which they were extracted. Unlike previous work in this area, our algorithm

is very general in its applicability, and scales well to large networks and problems with high-dimensional input spaces.

The task that we address is defined as follows: given a trained network and the data on which it was trained, produce a concept description that is comprehensible, yet classifies instances in the same way as the network. The concept description produced by our algorithm is a decision tree, like those generated using popular decision-tree induction algorithms (Breiman et al., 1984; Quinlan, 1993).

There are several reasons why the comprehensibility of induced concept descriptions is often an important consideration. If the designers and end-users of a learning system are to be confident in the performance of the system, they must understand how it arrives at its decisions. Learning systems may also play an important role in the process of scientific discovery. A system may discover salient features and relationships in the input data whose importance was not previously recognized. If the representations formed by the learner are comprehensible, then these discoveries can be made accessible to human review. However, for many problems in which comprehensibility is important, neural networks provide better generalization than common symbolic learning algorithms. It is in these domains that it is important to be able to extract comprehensible concept descriptions from trained networks.

## 2  Extracting Decision Trees

Our approach views the task of extracting a comprehensible concept description from a trained network as an inductive learning problem. In this learning task, the target concept is the function represented by the network, and the concept description produced by our learning algorithm is a decision tree that approximates the network. However, unlike most inductive learning problems, we have available an *oracle* that is able to answer queries during the learning process. Since the target function is simply the concept represented by the network, the oracle uses the network to answer queries. The advantage of learning with queries, as opposed to ordinary training examples, is that they can be used to garner information precisely where it is needed during the learning process.

Our algorithm, as shown in Table 1, is similar to conventional decision-tree algorithms, such as CART (Breiman et al., 1984), and C4.5 (Quinlan, 1993), which learn directly from a training set. However, TREPAN is substantially different from these conventional algorithms in number of respects, which we detail below.

**The Oracle.** The role of the oracle is to determine the class (as predicted by the network) of each instance that is presented as a query. Queries to the oracle, however, do not have to be complete instances, but instead can specify constraints on the values that the features can take. In the latter case, the oracle generates a complete instance by randomly selecting values for each feature, while ensuring that the constraints are satisfied. In order to generate these random values, TREPAN uses the training data to model each feature's marginal distribution. TREPAN uses frequency counts to model the distributions of discrete-valued features, and a kernel density estimation method (Silverman, 1986) to model continuous features. As shown in Table 1, the oracle is used for three different purposes: (i) to determine the class labels for the network's training examples; (ii) to select splits for each of the tree's internal nodes; (iii) and to determine if a node covers instances of only one class. These aspects of the algorithm are discussed in more detail below.

**Tree Expansion.** Unlike most decision-tree algorithms, which grow trees in a depth-first manner, TREPAN grows trees using a best-first expansion. The notion

Table 1: The TREPAN algorithm.

---

TREPAN(*training_examples, features*)
    *Queue* := ∅                                   /* sorted queue of nodes to expand */

    for each example $E \in$ *training_examples*                   /* use net to label examples */
        class label for $E$ := ORACLE($E$)

    initialize the root of the tree, $T$, as a leaf node
    put $\langle T,$ *training_examples*, {} $\rangle$ into *Queue*

    while *Queue* is not empty and size(T) < *tree_size_limit*      /* expand a node */
        remove node $N$ from head of *Queue*
        *examples$_N$* := example set stored with $N$
        *constraints$_N$* := constraint set stored with $N$

        use *features* to build set of candidate splits
        use *examples$_N$* and calls to ORACLE(*constraints$_N$*) to evaluate splits
        $S$ := best binary split
        search for best $m$-of-$n$ split, $S'$, using $S$ as a seed
        make $N$ an internal node with split $S'$

        for each outcome, $s$, of $S'$                       /* make children nodes */
            make $C$, a new child node of $N$
            *constraints$_C$* := *constraints$_N$* ∪ {$S' = s$}
            use calls to ORACLE(*constraints$_C$*) to determine if $C$ should remain a leaf
            otherwise
                *examples$_C$* := members of *examples$_N$* with outcome $s$ on split $S'$
                put $\langle C,$ *examples$_C$*, *constraints$_C$*$\rangle$ into *Queue*
    return $T$

---

of the best node, in this case, is the one at which there is the greatest potential to increase the fidelity of the extracted tree to the network. The function used to evaluate node $n$ is $f(n) = reach(n) \times (1 - fidelity(n))$, where $reach(n)$ is the estimated fraction of instances that reach $n$ when passed through the tree, and $fidelity(n)$ is the estimated fidelity of the tree to the network for those instances.

**Split Types.** The role of internal nodes in a decision tree is to partition the input space in order to increase the separation of instances of different classes. In C4.5, each of these splits is based on a single feature. Our algorithm, like Murphy and Pazzani's (1991) ID2-of-3 algorithm, forms trees that use $m$-of-$n$ expressions for its splits. An $m$-of-$n$ expression is a Boolean expression that is specified by an integer threshold, $m$, and a set of $n$ Boolean conditions. An $m$-of-$n$ expression is satisfied when at least $m$ of its $n$ conditions are satisfied. For example, suppose we have three Boolean features, $a$, $b$, and $c$; the $m$-of-$n$ expression $2$-$of$-$\{a, \neg b, c\}$ is logically equivalent to $(a \wedge \neg b) \vee (a \wedge c) \vee (\neg b \wedge c)$.

**Split Selection.** Split selection involves deciding how to partition the input space at a given internal node in the tree. A limitation of conventional tree-induction algorithms is that the amount of training data used to select splits decreases with the depth of the tree. Thus splits near the bottom of a tree are often poorly chosen because these decisions are based on few training examples. In contrast, because TREPAN has an oracle available, it is able to use as many instances as desired to select each split. TREPAN chooses a split after considering at least $S_{min}$ instances, where $S_{min}$ is a parameter of the algorithm.

When selecting a split at a given node, the oracle is given the list of all of the previously selected splits that lie on the path from the root of the tree to that node. These splits serve as constraints on the feature values that any instance generated by the oracle can take, since any example must satisfy these constraints in order to

reach the given node.

Like the ID2-of-3 algorithm, TREPAN uses a hill-climbing search process to construct its $m$-of-$n$ splits. The search process begins by first selecting the best binary split at the current node; as in C4.5, TREPAN uses the *gain ratio* criterion (Quinlan, 1993) to evaluate candidate splits. For two-valued features, a binary split separates examples according to their values for the feature. For discrete features with more than two values, we consider binary splits based on each allowable value of the feature (e.g., *color=red?, color=blue?, ...*). For continuous features, we consider binary splits on thresholds, in the same manner as C4.5. The selected binary split serves as a seed for the $m$-of-$n$ search process. This greedy search uses the *gain ratio* measure as its heuristic evaluation function, and uses the following two operators (Murphy & Pazzani, 1991):

- *m-of-n+1* : Add a new value to the set, and hold the threshold constant. For example, *2-of-{a, b}* $\Longrightarrow$ *2-of-{a, b, c}*.
- *m+1-of-n+1*: Add a new value to the set, and increment the threshold. For example, *2-of-{a, b, c}* $\Longrightarrow$ *3-of-{a, b, c, d}*.

Unlike ID2-of-3, TREPAN constrains $m$-of-$n$ splits so that the same feature is not used in two or more disjunctive splits which lie on the same path between the root and a leaf of the tree. Without this restriction, the oracle might have to solve difficult satisfiability problems in order create instances for nodes on such a path.

**Stopping Criteria.** TREPAN uses two separate criteria to decide when to stop growing an extracted decision tree. First, a given node becomes a leaf in the tree if, with high probability, the node covers only instances of a single class. To make this decision, TREPAN determines the proportion of examples, $p_c$, that fall into the most common class at a given node, and then calculates a confidence interval around this proportion (Hogg & Tanis, 1983). The oracle is queried for additional examples until $prob(p_c < 1 - \epsilon) < \delta$, where $\epsilon$ and $\delta$ are parameters of the algorithm.

TREPAN also accepts a parameter that specifies a limit on the number of internal nodes in an extracted tree. This parameter can be used to control the comprehensibility of extracted trees, since in some domains, it may require very large trees to describe networks to a high level of fidelity.

## 3  Empirical Evaluation

In our experiments, we are interested in evaluating the trees extracted by our algorithm according to three criteria: (i) their predictive accuracy; (ii) their comprehensibility; (i) and their fidelity to the networks from which they were extracted. We evaluate TREPAN using four real-world domains: the Congressional voting data set (15 features, 435 examples) and the Cleveland heart-disease data set (13 features, 303 examples) from the UC-Irvine database; a promoter data set (57 features, 468 examples) which is a more complex superset of the UC-Irvine one; and a data set in which the task is to recognize protein-coding regions in DNA (64 features, 20,000 examples) (Craven & Shavlik, 1993b). We remove the physician-fee-freeze feature from the voting data set to make the problem more difficult. We conduct our experiments using a 10-fold cross validation methodology, except for in the protein-coding domain. Because of certain domain-specific characteristics of this data set, we use 4-fold cross-validation for our experiments with it.

We measure accuracy and fidelity on the examples in the test sets. Whereas accuracy is defined as the percentage of test-set examples that are correctly classified, *fidelity* is defined as the percentage of test-set examples on which the classification

Table 2: Test-set accuracy and fidelity.

| domain | accuracy | | | | fidelity |
|---|---|---|---|---|---|
| | networks | C4.5 | ID2-of-3 | TREPAN | TREPAN |
| heart | 84.5% | 71.0% | 74.6% | 81.8% | 94.1% |
| promoters | 90.6 | 84.4 | 83.5 | 87.6 | 85.7 |
| protein coding | 94.1 | 90.3 | 90.9 | 91.4 | 92.4 |
| voting | 92.2 | 89.2 | 87.8 | 90.8 | 95.9 |

made by a tree agrees with its neural-network counterpart. Since the comprehensibility of a decision tree is problematic to measure, we measure the syntactic complexity of trees and take this as being representative of their comprehensibility. Specifically, we measure the complexity of each tree in two ways: (i) the number of internal (i.e., non-leaf) nodes in the tree, and (ii) the number of *symbols* used in the splits of the tree. We count an ordinary, single-feature split as one symbol. We count an $m$-of-$n$ split as $n$ symbols, since such a split lists $n$ feature values.

The neural networks we use in our experiments have a single layer of hidden units. The number of hidden units used for each network (0, 5, 10, 20 or 40) is chosen using cross validation on the network's training set, and we use a validation set to decide when to stop training networks. TREPAN is applied to each saved network. The parameters of TREPAN are set as follows for all runs: at least 1000 instances (training examples plus queries) are considered before selecting each split; we set the $\epsilon$ and $\delta$ parameters, which are used for the stopping-criterion procedure, to 0.05; and the maximum tree size is set to 15 internal nodes, which is the size of a complete binary tree of depth four.

As baselines for comparison, we also run Quinlan's (1993) C4.5 algorithm, and Murphy and Pazzani's (1991) ID2-of-3 algorithm on the same testbeds. Recall that ID2-of-3 is similar to C4.5, except that it learns trees that use $m$-of-$n$ splits. We use C4.5's pruning method for both algorithms and use cross validation to select pruning levels for each training set. The cross-validation runs evaluate unpruned trees and trees pruned with confidence levels ranging from 10% to 90%.

Table 2 shows the test-set accuracy results for our experiments. It can be seen that, for every data set, neural networks generalize better than the decision trees learned by C4.5 and ID2-of-3. The decision trees extracted from the networks by TREPAN are also more accurate than the C4.5 and ID2-of-3 trees in all domains. The differences in accuracy between the neural networks and the two conventional decision-tree algorithms (C4.5 and ID2-of-3) are statistically significant for all four domains at the 0.05 level using a paired, two-tailed $t$-test. We also test the significance of the accuracy differences between TREPAN and the other decision-tree algorithms. Except for the promoter domain, these differences are also statistically significant. The results in this table indicate that, for a range of interesting tasks, our algorithm is able to extract decision trees which are more accurate than decision trees induced strictly from the training data.

Table 2 also shows the test-set fidelity measurements for the TREPAN trees. These results indicate that the trees extracted by TREPAN provide close approximations to their respective neural networks.

Table 3 shows tree-complexity measurements for C4.5, ID2-of-3, and TREPAN. For all four data sets, the trees learned by TREPAN have fewer internal nodes than the trees produced by C4.5 and ID2-of-3. In most cases, the trees produced by TREPAN and ID2-of-3 use more symbols than C4.5, since their splits are more

Table 3: Tree complexity.

| domain | # internal nodes | | | # symbols | | |
|---|---|---|---|---|---|---|
| | C4.5 | ID2-of-3 | TREPAN | C4.5 | ID2-of-3 | TREPAN |
| heart | 17.5 | 15.7 | 11.8 | 17.5 | 48.8 | 20.8 |
| promoters | 11.2 | 12.6 | 9.2 | 11.2 | 47.5 | 23.8 |
| protein coding | 155.0 | 66.0 | 10.0 | 155.0 | 455.3 | 36.0 |
| voting | 20.1 | 19.2 | 11.2 | 20.1 | 77.3 | 20.8 |

complex. However, for most of the data sets, the TREPAN trees and the C4.5 trees are comparable in terms of their symbol complexity. For all data sets, the ID2-of-3 trees are more complex than the TREPAN trees. Based on these results, we argue that the trees extracted by TREPAN are as comprehensible as the trees learned by conventional decision-tree algorithms.

## 4  Discussion and Conclusions

In the previous section, we evaluated our algorithm along the dimensions of fidelity, syntactic complexity, and accuracy. Another advantage of our approach is its generality. Unlike numerous other extraction methods (Hayashi, 1991; McMillan et al., 1992; Craven & Shavlik, 1993a; Sethi et al., 1993; Tan, 1994; Tchoumatchenko & Ganascia, 1994; Alexander & Mozer, 1995; Setiono & Liu, 1995), the TREPAN algorithm does not place any requirements on either the architecture of the network or its training method. TREPAN simply uses the network as a black box to answer queries during the extraction process. In fact, TREPAN could be used to extract decision-trees from other types of opaque learning systems, such as nearest-neighbor classifiers.

There are several existing algorithms which do not require special network architectures or training procedures (Saito & Nakano, 1988; Fu, 1991; Gallant, 1993). These algorithms, however, assume that each hidden unit in a network can be accurately approximated by a threshold unit. Additionally, these algorithms do not extract $m$-of-$n$ rules, but instead extract only conjunctive rules. In previous work (Craven & Shavlik, 1994; Towell & Shavlik, 1993), we have shown that this type of algorithm produces rule-sets which typically are far too complex to be comprehensible. Thrun (1995) has developed a general method for rule extraction, and has described how his algorithm can be used to *verify* that an $m$-of-$n$ rule is consistent with a network, but he has not developed a rule-searching method that is able to find concise rule sets. A strength of our algorithm, in contrast, is its scalability. We have demonstrated that our algorithm is able to produce succinct decision-tree descriptions of large networks in domains with large input spaces.

In summary, a significant limitation of neural networks is that their concept representations are usually not amenable to human understanding. We have presented an algorithm that is able to produce comprehensible descriptions of trained networks by extracting decision trees that accurately describe the networks' concept representations. We believe that our algorithm, which takes advantage of the fact that a trained network can be queried, represents a promising advance towards the goal of general methods for understanding the solutions encoded by trained networks.

### Acknowledgements

This research was partially supported by ONR grant N00014-93-1-0998.

# References

Alexander, J. A. & Mozer, M. C. (1995). Template-based algorithms for connectionist rule extraction. In Tesauro, G., Touretzky, D., & Leen, T., editors, *Advances in Neural Information Processing Systems (volume 7)*. MIT Press.

Breiman, L., Friedman, J., Olshen, R., & Stone, C. (1984). *Classification and Regression Trees*. Wadsworth and Brooks, Monterey, CA.

Craven, M. & Shavlik, J. (1993a). Learning symbolic rules using artificial neural networks. In *Proc. of the 10th International Conference on Machine Learning*, (pp. 73–80), Amherst, MA. Morgan Kaufmann.

Craven, M. W. & Shavlik, J. W. (1993b). Learning to predict reading frames in *E. coli* DNA sequences. In *Proc. of the 26th Hawaii International Conference on System Sciences*, (pp. 773–782), Wailea, HI. IEEE Press.

Craven, M. W. & Shavlik, J. W. (1994). Using sampling and queries to extract rules from trained neural networks. In *Proc. of the 11th International Conference on Machine Learning*, (pp. 37–45), New Brunswick, NJ. Morgan Kaufmann.

Fu, L. (1991). Rule learning by searching on adapted nets. In *Proc. of the 9th National Conference on Artificial Intelligence*, (pp. 590–595), Anaheim, CA. AAAI/MIT Press.

Gallant, S. I. (1993). *Neural Network Learning and Expert Systems*. MIT Press.

Hayashi, Y. (1991). A neural expert system with automated extraction of fuzzy if-then rules. In Lippmann, R., Moody, J., & Touretzky, D., editors, *Advances in Neural Information Processing Systems (volume 3)*. Morgan Kaufmann, San Mateo, CA.

Hogg, R. V. & Tanis, E. A. (1983). *Probability and Statistical Inference*. MacMillan.

McMillan, C., Mozer, M. C., & Smolensky, P. (1992). Rule induction through integrated symbolic and subsymbolic processing. In Moody, J., Hanson, S., & Lippmann, R., editors, *Advances in Neural Information Processing Systems (volume 4)*. Morgan Kaufmann.

Murphy, P. M. & Pazzani, M. J. (1991). ID2-of-3: Constructive induction of M-of-N concepts for discriminators in decision trees. In *Proc. of the 8th International Machine Learning Workshop*, (pp. 183–187), Evanston, IL. Morgan Kaufmann.

Quinlan, J. (1993). *C4.5: Programs for Machine Learning*. Morgan Kaufmann.

Saito, K. & Nakano, R. (1988). Medical diagnostic expert system based on PDP model. In *Proc. of the IEEE International Conference on Neural Networks*, (pp. 255–262), San Diego, CA. IEEE Press.

Sethi, I. K., Yoo, J. H., & Brickman, C. M. (1993). Extraction of diagnostic rules using neural networks. In *Proc. of the 6th IEEE Symposium on Computer-Based Medical Systems*, (pp. 217–222), Ann Arbor, MI. IEEE Press.

Setiono, R. & Liu, H. (1995). Understanding neural networks via rule extraction. In *Proc. of the 14th International Joint Conference on Artificial Intelligence*, (pp. 480–485), Montreal, Canada.

Silverman, B. W. (1986). *Density Estimation for Statistics and Data Analysis*. Chapman and Hall.

Tan, A.-H. (1994). Rule learning and extraction with self-organizing neural networks. In *Proc. of the 1993 Connectionist Models Summer School*. Erlbaum.

Tchoumatchenko, I. & Ganascia, J.-G. (1994). A Bayesian framework to integrate symbolic and neural learning. In *Proc. of the 11th International Conference on Machine Learning*, (pp. 302–308), New Brunswick, NJ. Morgan Kaufmann.

Thrun, S. (1995). Extracting rules from artificial neural networks with distributed representations. In Tesauro, G., Touretzky, D., & Leen, T., editors, *Advances in Neural Information Processing Systems (volume 7)*. MIT Press.

Towell, G. & Shavlik, J. (1993). Extracting refined rules from knowledge-based neural networks. *Machine Learning*, 13(1):71–101.
